# Worst-Case Bounds for Gaussian Process Models

**Sham M. Kakade**
University of Pennsylvania

**Matthias W. Seeger**
UC Berkeley

**Dean P. Foster**
University of Pennsylvania

## Abstract

We present a competitive analysis of some non-parametric Bayesian algorithms in a worst-case online learning setting, where no probabilistic assumptions about the generation of the data are made. We consider models which use a Gaussian process prior (over the space of all functions) and provide bounds on the regret (under the log loss) for commonly used non-parametric Bayesian algorithms — including Gaussian regression and logistic regression — which show how these algorithms can perform favorably under rather general conditions. These bounds explicitly handle the infinite dimensionality of these non-parametric classes in a natural way. We also make formal connections to the minimax and *minimum description length* (MDL) framework. Here, we show precisely how Bayesian Gaussian regression is a minimax strategy.

## 1 Introduction

We study an online (sequential) prediction setting in which, at each timestep, the learner is given some input from the set $\mathcal{X}$, and the learner must predict the output variable from the set $\mathcal{Y}$. The sequence $\{(x_t, y_t) \mid t = 1, \ldots, T\}$ is chosen by Nature (or by an adversary), and importantly, we do not make any statistical assumptions about its source: our statements hold for *all* sequences. Our goal is to sequentially code the next label $y_t$, given that we have observed $\boldsymbol{x}_{\leq t}$ and $\boldsymbol{y}_{<t}$ (where $\boldsymbol{x}_{\leq t}$ and $\boldsymbol{y}_{<t}$ denote the sequences $\{x_1, \ldots x_t\}$ and $\{y_1, \ldots y_{t-1}\}$). At each time $t$, we have a conditional distribution $P(\cdot | \boldsymbol{x}_{\leq t}, \boldsymbol{y}_{<t})$ over $\mathcal{Y}$, which is our prediction strategy that is used to predict the next variable $y_t$. We then incur the instantaneous loss $-\log P(y_t | \boldsymbol{x}_{\leq t}, \boldsymbol{y}_{<t})$ (referred to as *log loss*), and the cumulative loss is the sum of these instantaneous losses over $t = 1, \ldots, T$.

Let $\Theta$ be a parameter space indexing elementary prediction rules in some model class, where $P(y | x, \theta)$ for $\theta \in \Theta$ is a conditional distribution over $\mathcal{Y}$ called the *likelihood*. An *expert* is a single atom $\theta \in \Theta$, or, more precisely, the algorithm which outputs the predictive distribution $P(\cdot | x_t, \theta)$ for every $t$. We are interested in bounds on the *regret* — the difference in the cumulative loss of a given adaptive prediction strategy and the the cumulative loss of the best possible expert chosen in hindsight from a subset of $\Theta$.

Kakade and Ng [2004] considered a parametric setting where $\Theta = \mathbb{R}^d$, $\mathcal{X} = \mathbb{R}^d$, and the prediction rules were generalized linear models, in which $P(y | x, \theta) = P(y | \theta \cdot x)$. They derived regret bounds for the Bayesian strategy (assuming a Gaussian prior over $\Theta$), which showed that many simple Bayesian algorithms (such as Gaussian linear regression and logistic regression) perform favorably when compared, in retrospect, to the best $\theta \in \Theta$. Importantly, these regret bounds have a time and dimensionality dependence of the form $\frac{d}{2} \log T$ — a dependence common in in most MDL procedures (see Grunwald [2005]). For Gaussian linear regression, the bounds of Kakade and Ng [2004] are comparable to the best bounds in the literature, such as those of Foster [1991], Vovk [2001], Azoury and Warmuth

[2001] (though these latter bounds are stated in terms of the closely related square loss).

In this paper, we provide worst-case regret bounds on Bayesian non-parametric methods, which show how these algorithms can have low regret. In particular, we examine the case where the prior (over functions) is a Gaussian process — thereby extending the work of Kakade and Ng [2004] to infinite-dimensional spaces of experts. There are a number of important differences between this and the parametric setting. First, it turns out that the natural competitor class is the *reproducing kernel Hilbert space (RKHS)* $\mathcal{H}$. Furthermore, the notion of dimensionality is more subtle, since the space $\mathcal{H}$ may be infinite dimensional. In general, there is no apriori reason that any strategy (including the Bayesian one) should be able to compete favorably with the complex class $\mathcal{H}$. However, for some input sequences $\boldsymbol{x}_{\leq T}$ and kernels, we show that it is possible to compete favorably. Furthermore, the relation of our results to Kakade and Ng [2004] is made explicit in Section 3.2.

Our second contribution is in making formal connections to minimax theory, where we show precisely how Bayesian Gaussian regression is a minimax algorithm. In a general setting, Shtarkov [1987] showed that a certain *normalized maximum likelihood* (NML) distribution minimizes the regret in the worst case. Unfortunately, for some "complex" model classes, there may exist no strategy which achieves finite regret, and so the NML distribution may not exist.[1] Gaussian density estimation (formally described in Example 4.2) is one such case where this NML distribution does not exist. If one makes further restrictions (on $\mathcal{Y}$), then minimax results can be derived, such as in Takimoto and Warmuth [2000], Barron et al. [1998], Foster and Stine [2001].

Instead of making further restrictions, we propose minimizing a form of a *penalized regret*, where one penalizes more "complex" experts as measured by their cost under a prior $q(\theta)$. This penalized regret essentially compares our cumulative loss to the loss of a two part code (common in MDL, see Grunwald [2005]), where one first codes the model $\theta$ under a prior $q$ and then codes the data using this $\theta$. Here, we show that a certain *normalized maximum a posteriori* distribution is the corresponding minimax strategy, in general. Our main result here is in showing that for Gaussian regression, the Bayesian strategy is precisely this minimax strategy. The differences between this result and that of Takimoto and Warmuth [2000] are notable. In the later, they assume $\mathcal{Y} \subset \mathbb{R}$ is bounded and derive (near) minimax algorithms which hold the variance of their predictions *constant* at each timestep (so they effectively deal with the square loss). Under Bayes rule, the variance of the predictions adapts, which allows the minimax property to hold with $\mathcal{Y} = \mathbb{R}$ being unbounded.

Other minimax results have been considered in the non-parametric setting. The work of Opper and Haussler [1998] and Cesa-Bianchi and Lugosi [2001] provide minimax bounds in some non-parametric cases (in terms of a covering number of the comparator class), though they do not consider input sequences.

The rest of the paper is organized as follows: Section 2 summarizes our model, Section 3 presents and discusses our bounds, and Section 4 draws out the connections to the minimax and MDL framework. All proofs are available in a forthcoming longer version of this paper.

## 2   Bayesian Methods with Gaussian Process Priors

With a Bayesian prior distribution $P_{\text{bayes}}(\theta)$ over $\Theta$, the Bayesian predicts $y_t$ using the rule

$$P_{\text{bayes}}(y_t | \boldsymbol{x}_{\leq t}, \boldsymbol{y}_{<t}) = \int P(y_t | x_t, \theta) P_{\text{bayes}}(\theta | \boldsymbol{x}_{<t}, \boldsymbol{y}_{<t}) \, d\theta$$

where the posterior is given by

$$P_{\text{bayes}}(\theta | \boldsymbol{x}_{<t}, \boldsymbol{y}_{<t}) \propto P(\boldsymbol{y}_{<t} | \boldsymbol{x}_{<t}, \theta) P_{\text{bayes}}(\theta).$$

Assuming the Bayesian learner models the data to be independent given $\theta$, then

$$P(\boldsymbol{y}_{<t}|\boldsymbol{x}_{<t},\theta) = \prod_{t'=1}^{t-1} P(y_{t'}|x_{t'},\theta)\,.$$

It is important to stress that these are "working assumptions" in the sense that they lead to a prediction strategy (the Bayesian one), but the analysis does *not* make any probabilistic assumptions about the generation of the data. The cumulative loss of the Bayesian strategy is then

$$-\sum_{t=1}^{T} \log P_{\text{bayes}}(y_t|\boldsymbol{x}_{\leq t},\boldsymbol{y}_{<t}) = -\log P_{\text{bayes}}(\boldsymbol{y}_{\leq T}|\boldsymbol{x}_{\leq T}).$$

which follows form the chain rule of conditional probabilities.

In this paper, we are interested in non-parametric prediction, which can be viewed as working with an infinite-dimensional function space $\Theta$ — assume $\Theta$ consists of real-valued functions $u(x)$. The likelihood $P(y|x,u(\cdot))$ is thus a distribution over $y$ given $x$ and the function $u(\cdot)$. Similar to Kakade and Ng [2004] (where they considered generalized linear models), we make the natural restriction that $P(y|x,u(\cdot)) = P(y|u(x))$. We can think of $u$ as a latent function and of $P(y|u(x))$ as a noise distribution. Two particularly important cases are that of Gaussian regression and logistic regression. In *Gaussian regression*, we have that $\mathcal{Y} = \mathbb{R}$ and that $P(y|u(x)) = \mathcal{N}(y|u(x),\sigma^2)$ (so $y$ is distributed as a Gaussian with mean $u(x)$ and fixed variance $\sigma^2$). In *logistic regression*, $\mathcal{Y} = \{-1,1\}$ and $P(y|u(x)) = (1 + e^{-yu(x)})^{-1}$.

In this paper, we consider the case in which the prior $dP_{\text{bayes}}(u(\cdot))$ is a zero-mean *Gaussian process (GP)* with covariance function $K$, *i.e.* a real-valued random process which has the property that for every finite set $x_1,\ldots,x_n$ the random vector $(u(x_1),\ldots,u(x_n))^T$ is multivariate Gaussian, distributed as $\mathcal{N}(\boldsymbol{0},\boldsymbol{K})$, where $\boldsymbol{K} \in \mathbb{R}^{n,n}$ is the covariance (or kernel) matrix with $\boldsymbol{K}_{i,j} = K(x_i,x_j)$. Note that $K$ has to be a positive semidefinite function in that for all finite sets $x_1,\ldots,x_n$ the corresponding kernel matrices $\boldsymbol{K}$ are positive semidefinite.

Finally, we specify the subset of experts we would like the Bayesian prediction strategy to compete against. Every positive semidefinite kernel $K$ is associated with a unique *reproducing kernel Hilbert space (RKHS)* $\mathcal{H}$, defined as follows: consider the linear space of all finite kernel expansions (over any $x_1,\ldots,x_n$) of the form $f(x) = \sum_{i=1}^{n} \alpha_i K(x,x_i)$ with the inner product

$$\left(\sum_i \alpha_i K(\cdot,x_i), \sum_j \beta_j K(\cdot,y_j)\right)_K = \sum_{i,j} \alpha_i \beta_j K(x_i,y_j).$$

and define the RKHS $\mathcal{H}$ as the completion of this space. By construction, $\mathcal{H}$ contains all finite kernel expansions $f(x) = \sum_{i=1}^{n} \alpha_i K(x,x_i)$ with

$$\|f\|_K^2 = \boldsymbol{\alpha}^T \boldsymbol{K} \boldsymbol{\alpha}, \quad \boldsymbol{K}_{i,j} = K(x_i,x_j)\,. \tag{1}$$

The characteristic property of $\mathcal{H}$ is that all (Dirac) evaluation functionals are *represented* in $\mathcal{H}$ itself by the functions $K(\cdot,x_i)$, meaning $(f,K(\cdot,x_i))_K = f(x_i)$. The RKHS $\mathcal{H}$ turns out to be the largest subspace of experts for which our results are meaningful.

## 3   Worst-Case Bounds

In this section, we present our worst-case bounds, give an interpretation, and relate the results to the parametric case of Kakade and Ng [2004]. The proofs are available in a forthcoming longer version.

**Theorem 3.1:** *Let $(\boldsymbol{x}_{\leq T}, \boldsymbol{y}_{\leq T})$ be a sequence from $(\mathcal{X} \times \mathcal{Y})^T$. For all functions $f$ in the RKHS $\mathcal{H}$ associated with the prior covariance function $K$, we have*

$$-\log P_{bayes}(\boldsymbol{y}_{\leq T}|\boldsymbol{x}_{\leq T}) \leq -\log P(\boldsymbol{y}_{\leq T}|\boldsymbol{x}_{\leq T}, f(\cdot)) + \frac{1}{2}\|f\|_K^2 + \frac{1}{2}\log|\boldsymbol{I} + c\boldsymbol{K}|,$$

*where $\|f\|_K$ is the RKHS norm of $f$, $\boldsymbol{K} = (K(x_t, x_{t'})) \in \mathbb{R}^{T,T}$ is the kernel matrix over the input sequence $\boldsymbol{x}_{\leq T}$, and $c > 0$ is a constant such that for all $y_t \in \boldsymbol{y}_{\leq T}$,*

$$-\frac{d^2}{du^2}\log P(y_t|u) \leq c$$

*for all $u \in \mathbb{R}$.*

The proof of this theorem parallels that provided by Kakade and Ng [2004], with a number of added complexities for handling GP priors. For the special case of Gaussian regression where $c = \sigma^{-2}$, the following theorem shows the stronger result that the bound is satisfied with an equality for all sequences.

**Theorem 3.2:** *Assume $P(y_t|u(x_t)) = \mathcal{N}(y_t|u(x_t), \sigma^2)$ and that $\mathcal{Y} = \mathbb{R}$. Let $(\boldsymbol{x}_{\leq T}, \boldsymbol{y}_{\leq T})$ be a sequence from $(\mathcal{X} \times \mathcal{Y})^T$. Then,*

$$-\log P_{bayes}(\boldsymbol{y}_{\leq T}|\boldsymbol{x}_{\leq T}) = \min_{f \in \mathcal{H}} \left\{ -\log P(\boldsymbol{y}_{\leq T}|\boldsymbol{x}_{\leq T}, f(\cdot)) + \frac{1}{2}\|f\|_K^2 \right\} \tag{2}$$
$$+ \frac{1}{2}\log|\boldsymbol{I} + \sigma^{-2}\boldsymbol{K}|$$

*and the minimum is attained for a kernel expansion over $\boldsymbol{x}_{\leq T}$.*

This equality has important implications in our minimax theory (in Corollary 4.4, we make this precise). It is not hard to see that the equality does not hold for other likelihoods.

## 3.1 Interpretation

The regret bound depends on two terms, $\|f\|_K^2$ and $\log|\boldsymbol{I} + c\boldsymbol{K}|$. We discuss each in turn. The dependence on $\|f\|_K^2$ states the intuitive fact that a meaningful bound can only be obtained under smoothness assumptions on the set of experts. The more complicated $f$ is (as measured by $\|\cdot\|_K$), the higher the regret may be. The equality shows in Theorem 3.2 shows this dependence is unavoidable. We come back to this dependence in Section 4.

Let us now interpret the $\log|\boldsymbol{I} + c\boldsymbol{K}|$ term, which we refer to as the regret term. The constant $c$, which bounds the curvature of the likelihood, exists for most commonly used exponential family likelihoods. For logistic regression, we have $c = 1/4$, and for the Gaussian regression, we have $c = \sigma^{-2}$. Also, interestingly, while $f$ is an arbitrary function in $\mathcal{H}$, this regret term depends on $\boldsymbol{K}$ only at the sequence points $\boldsymbol{x}_{\leq T}$.

For most infinite-dimensional kernels and without strong restrictions on the inputs, the regret term can be as large as $\Omega(T)$ — the sequence can be chosen s.t. $\boldsymbol{K} \approx c'\boldsymbol{I}$, which implies that $\log|\boldsymbol{I} + c\boldsymbol{K}| \approx T\log(1 + cc')$. For example, for an isotropic kernel (which is a function of the norm $\|x - x'\|_2$) we can choose the $x_t$ to be mutually far from each other. For kernels which barely enforce smoothness — e.g. the Ornstein-Uhlenbeck kernel $\exp(-b\|x - x'\|_1)$ — the regret term can easily $\Omega(T)$. The cases we are interested in are those where the regret term is $o(T)$, in which case the average regret tends to 0 with time.

A spectral interpretation of this term helps us understand the behavior. If we let the $\lambda_1, \lambda_2, \ldots \lambda_T$ be the eigenvalues of $\boldsymbol{K}$, then

$$\log|\boldsymbol{I} + c\boldsymbol{K}| = \sum_{t=1}^{T} \log(1 + c\lambda_t) \leq c\,\mathrm{tr}\,\boldsymbol{K}$$

where $\operatorname{tr} \boldsymbol{K}$ is the trace of $\boldsymbol{K}$. This last quantity is closely related to the "degrees of freedom" in a system (see Hastie et al. [2001]). Clearly, if the sum of the eigenvalues has a sublinear growth rate of $o(T)$, then the average regret tends to $0$. Also, if one assumes that the input sequence, $\boldsymbol{x}_{\leq T}$, is i.i.d. then the above eigenvalues are essentially the *process* eigenvalues. In a forthcoming longer version, we explore this spectral interpretation in more detail and provide a case using the exponential kernel in which the regret grows as $O(\operatorname{poly}(\log T))$. We now review the parametric case.

### 3.2 The Parametric Case

Here we obtain a slight generalization of the result in Kakade and Ng [2004] as a special case. Namely, the familiar linear model — with $u(x) = \theta \cdot x$, $\theta, x \in \mathbb{R}^d$ and Gaussian prior $\theta \sim \mathcal{N}(\mathbf{0}, \boldsymbol{I})$ — can be seen as a GP model with the linear kernel: $K(x, x') = x \cdot x'$.

With $\boldsymbol{X} = (x_1, \ldots x_T)^{\mathrm{T}}$ we have that a kernel expansion $f(x) = \sum_i \alpha_i x_i \cdot x = \theta \cdot x$ with $\theta = \boldsymbol{X}^{\mathrm{T}} \boldsymbol{\alpha}$, and $\|f\|_K^2 = \boldsymbol{\alpha}^{\mathrm{T}} \boldsymbol{X} \boldsymbol{X}^{\mathrm{T}} \boldsymbol{\alpha} = \|\theta\|_2^2$, so that $\mathcal{H} = \{\theta \cdot x \,|\, \theta \in \mathbb{R}^d\}$, and so

$$\log |\boldsymbol{I} + c\boldsymbol{K}| = \log \left| \boldsymbol{I} + c\boldsymbol{X}^{\mathrm{T}}\boldsymbol{X} \right|$$

Therefore, our result gives an input-dependent version of the result of Kakade and Ng [2004]. If we make the further assumption that $\|x\|_2 \leq 1$ (as done in Kakade and Ng [2004]), then we can obtain exactly their regret term:

$$\log |\boldsymbol{I} + c\boldsymbol{K}| \leq d \log \left( 1 + \frac{cT}{d} \right)$$

which can seen by rotating $K$ into an diagonal matrix and maximizing the expression subject to the constraint that $\|x\|_2 \leq 1$ (i.e. that the eigenvalues must sum to 1).

In general, this example shows that if $K$ is a finite-dimension kernel such as the linear or the polynomial kernel, then the regret term is only $O(\log T)$.

## 4 Relationships to Minimax Procedures and MDL

This section builds the framework for understanding the minimax property of Gaussian regression. We start by reviewing Shtarkov's theorem, which shows that a certain normalized maximum likelihood density is the minimax strategy (when using the log loss). In many cases, this minimax strategy does not exist — in those cases where the minimax regret is infinite. We then propose a different, penalized notion of regret, and show that a certain *normalized maximum a posteriori* density is the minimax strategy here. Our main result (Corollary 4.4) shows that for Gaussian regression the Bayesian strategy is precisely this minimax strategy

### 4.1 Normalized Maximum Likelihood

Here, let us assume that there are no inputs — sequences consist of only $y_t \in \mathcal{Y}$. Given a measurable space with base measure $\mu$, we employ a countable number of random variables $y_t$ in $\mathcal{Y}$. Fix the sequence length $T$ and define the model class $\mathcal{F} = \{Q(\cdot|\theta) \,|\, \theta \in \Theta\}$, where $Q(\cdot|\theta)$ denotes a joint probability density over $\mathcal{Y}^T$ with respect to $\mu$.

We assume that for our model class there exists a parameter, $\theta_{\mathrm{ml}}(\boldsymbol{y}_{\leq T})$, maximizing the likelihood $Q(\boldsymbol{y}_{\leq T}|\theta)$ over $\Theta$ for all $\boldsymbol{y}_{\leq T} \in \mathcal{Y}^T$. We make this assumption to make the connections to maximum likelihood (and, later, MAP) estimation clear. Define the regret of a joint density $P$ on $\boldsymbol{y}_{\leq T}$ as:

$$\begin{aligned} R(\boldsymbol{y}_{\leq T}, P, \Theta) &= -\log P(\boldsymbol{y}_{\leq T}) - \inf_{\theta \in \Theta} \{-\log Q(\boldsymbol{y}_{\leq T}|\theta)\} && (3) \\ &= -\log P(\boldsymbol{y}_{\leq T}) + \log Q(\boldsymbol{y}_{\leq T}|\theta_{\mathrm{ml}}(\boldsymbol{y}_{\leq T})) && (4) \end{aligned}$$

where the latter step uses our assumption on the existence of $\theta_{\mathrm{ml}}(\boldsymbol{y}_{\leq T})$.

Define the minimax regret with respect to $\Theta$ as:

$$R(\Theta) = \inf_{P} \sup_{\boldsymbol{y}_{\leq T} \in \mathcal{Y}^T} R(\boldsymbol{y}_{\leq T}, P, \Theta)$$

where the $\inf$ is over all probability densities on $\mathcal{Y}^T$.

The following theorem due to Shtarkov [1987] characterizes the minimax strategy.

**Theorem 4.1:** *[Shtarkov, 1987]If the following density exists (i.e. if it has a finite normalization constant), then define it to be the* normalized maximum likelihood *(NML) density.*

$$P_{ml}(\boldsymbol{y}_{\leq T}) = \frac{Q(\boldsymbol{y}_{\leq T}|\theta_{ml}(\boldsymbol{y}_{\leq T}))}{\int Q(\boldsymbol{y}_{\leq T}|\theta_{ml}(\boldsymbol{y}_{\leq T}))d\mu(\boldsymbol{y}_{\leq T})} \tag{5}$$

*If $P_{ml}$ exists, it is a minimax strategy, i.e. for all $\boldsymbol{y}_{\leq T}$, the regret $R(\boldsymbol{y}_{\leq T}, P_{ml}, \Theta)$ does not exceed $R(\Theta)$.*

Note that this density exists only if the normalizing constant is finite, which is not the case in general. The proof is straightforward using the fact that the NML density is an *equalizer* — meaning that it has *constant* regret on all sequences.

**Proof:** First note that the regret $R(\boldsymbol{y}_{\leq T}, P_{\mathrm{ml}}, \Theta)$ is the constant $\log \int Q(\boldsymbol{y}_{\leq T}|\theta_{\mathrm{ml}}(\boldsymbol{y}_{\leq T}))d\mu(\boldsymbol{y}_{\leq T})$. To see this, simply substitute Eq. 5 into Eq. 4 and simplify.

For convenience, define the regret of any $P$ as $R(P, \Theta) = \sup_{\boldsymbol{y}_{\leq T} \in \mathcal{Y}^T} R(\boldsymbol{y}_{\leq T}, P, \Theta)$. For any $P \neq P_{\mathrm{ml}}$ (differing on a set with positive measure), there exists some $\boldsymbol{y}_{\leq T}$ such that $P(\boldsymbol{y}_{\leq T}) < P_{\mathrm{ml}}(\boldsymbol{y}_{\leq T})$, since the densities are normalized. This implies that

$$R(P, \Theta) \geq R(\boldsymbol{y}_{\leq T}, P, \Theta) > R(\boldsymbol{y}_{\leq T}, P_{\mathrm{ml}}, \Theta) = R(P_{\mathrm{ml}}, \Theta)$$

where the first step follows from the definition of $R(P, \Theta)$, the second from $-\log P(\boldsymbol{y}_{\leq T}) > -\log P_{\mathrm{ml}}(\boldsymbol{y}_{\leq T})$, and the last from the fact that $P_{\mathrm{ml}}$ is an equalizer (its regret is constant on all sequences). Hence, $P$ has a strictly larger regret, implying that $P_{\mathrm{ml}}$ is the unique minimax strategy. $\square$

Unfortunately, in many important model classes, the minimax regret $R(\Theta)$ is not finite, and the NML density does not exist. We now provide one example (see Grunwald [2005] for further discussion).

**Example 4.2:** Consider a model which assumes the sequence is generated i.i.d. from a Gaussian with unknown mean and unit variance. Specifically, let $\Theta = \mathbb{R}$, $\mathcal{Y} = R$, and $P(\boldsymbol{y}_{\leq T}|\theta)$ be the product $\Pi_{t=1}^{T}\mathcal{N}(y_t; \theta, 1)$. It is easy to see that for this class the minimax regret is infinite and $P_{\mathrm{ml}}$ does not exist (see Grunwald [2005]). This example can be generalized to the Gaussian regression model (if we know the sequence $\boldsymbol{x}_{\leq T}$ in advance). For this problem, if one modifies the space of allowable sequences (i.e. $\bar{\mathcal{Y}}^T$ is modified), then one can obtain finite regret, such as those in Barron et al. [1998], Foster and Stine [2001]. This technique may not be appropriate in general.

## 4.2 Normalized Maximum a Posteriori

To remedy this problem, consider placing some structure on the model class $\mathcal{F} = \{Q(\cdot|\theta)|\theta \in \Theta\}$. The idea is to penalize $Q(\cdot|\theta) \in \mathcal{F}$ based on this structure. The motivation is similar to that of structural risk minimization [Vapnik, 1998]. Assume that $\Theta$ is

a measurable space and place a prior distribution with density function $q$ on $\Theta$. Define the *penalized regret* of $P$ on $\boldsymbol{y}_{\leq T}$ as:

$$R_q(\boldsymbol{y}_{\leq T}, P, \Theta) = -\log P(\boldsymbol{y}_{\leq T}) - \inf_{\theta \in \Theta} \{-\log Q(\boldsymbol{y}_{\leq T}|\theta) - \log q(\theta)\}.$$

Note that $-\log Q(\boldsymbol{y}_{\leq T}|\theta) - \log q(\theta)$ can be viewed as a "two part" code, in which we first code $\theta$ under the prior $q$ and then code $\boldsymbol{y}_{\leq T}$ under the likelihood $Q(\cdot|\theta)$. Unlike the standard regret, the penalized regret can be viewed as a comparison to an actual code. These two part codes are common in the MDL literature (see Grunwald [2005]). However, in MDL, they consider using minimax schemes (via $P_{\mathrm{ml}}$) for the likelihood part of the code, while we consider minimax schemes for this penalized regret.

Again, for clarity, assume there exists a parameter, $\theta_{\mathrm{map}}(\boldsymbol{y}_{\leq T})$ maximizing $\log Q(\boldsymbol{y}_{\leq T}|\theta) + \log q(\theta)$. Notice that this is just the maximum aposteriori (MAP) parameter, if one were to use a Bayesian strategy with the prior $q$ (since the posterior density would be proportional to $Q(\boldsymbol{y}_{\leq T}|\theta)q(\theta)$). Here,

$$R_q(\boldsymbol{y}_{\leq T}, P, \Theta) = -\log P(\boldsymbol{y}_{\leq T}) + \log Q(\boldsymbol{y}_{\leq T}|\theta_{\mathrm{map}}(\boldsymbol{y}_{\leq T})) + \log q(\theta_{\mathrm{map}}(\boldsymbol{y}_{\leq T}))$$

Similarly, with respect to $\Theta$, define the minimax penalized regret as:

$$R_q(\Theta) = \inf_P \sup_{\boldsymbol{y}_{\leq T} \in \mathcal{Y}^T} R_q(\boldsymbol{y}_{\leq T} P, \Theta)$$

where again the $\inf$ is over all densities on $\mathcal{Y}^T$. If $\Theta$ is finite or countable and $Q(\cdot|\theta) > 0$ for all $\theta$, then the Bayes procedure has the desirable property of having penalized regret which is non-positive.[2] However, in general, the Bayes procedure does not achieve the minimax penalized regret, $R_q(\Theta)$, which is what we desire — though, for one case, we show that it does (in the next section).

We now characterize this minimax strategy in general.

**Theorem 4.3:** *Define the* normalized maximum a posteriori *(NMAP) density, if it exists, as:*

$$P_{map}(\boldsymbol{y}_{\leq T}) = \frac{Q(\boldsymbol{y}_{\leq T}|\theta_{map}(\boldsymbol{y}_{\leq T}))q(\theta_{map}(\boldsymbol{y}_{\leq T}))}{\int Q(\boldsymbol{y}_{\leq T}|\theta_{map}(\boldsymbol{y}_{\leq T}))q(\theta_{map}(\boldsymbol{y}_{\leq T}))\,d\mu(\boldsymbol{y}_{\leq T})}. \tag{6}$$

*If $P_{map}$ exists, it is a minimax strategy for the penalized regret, i.e. for all $\boldsymbol{y}_{\leq T}$, the penalized regret $R_q(\boldsymbol{y}_{\leq T}, P_{map}, \Theta)$ does not exceed $R_q(\Theta)$.*

The proof relies on $P_{\mathrm{map}}$ being an *equalizer* for the penalized regret and is identical to that of Theorem 4.1 — just replace all quantities with their penalized equivalents.

### 4.3 Bayesian Gaussian Regression as a Minimax Procedure

We now return to the setting with inputs and show how the Bayesian strategy for the Gaussian regression model is a minimax strategy *for all input sequences* $\boldsymbol{x}_{\leq T}$. If we fix the input sequence $\boldsymbol{x}_{\leq T}$, we can consider the competitor class to be $\mathcal{F} = \{\overline{P}(\boldsymbol{y}_{\leq T}|\boldsymbol{x}_{\leq T}, \theta) \,|\, \theta \in \Theta)\}$. In other words, we make the more stringent comparison against a model class which has *full knowledge* of the input sequence in advance. Importantly, note that the learner only observes the past inputs $\boldsymbol{x}_{<t}$ at time $t$.

Consider the Gaussian regression model, with likelihood $P(\boldsymbol{y}_{\leq T}|\boldsymbol{x}_{\leq T}, u(\cdot)) = \mathcal{N}(\boldsymbol{y}_{\leq T}|u(\boldsymbol{x}_{\leq T}), \sigma^2 \boldsymbol{I})$, where $u(\cdot)$ is some function and $\boldsymbol{I}$ is the $T \times T$ identity. For

technical reasons, we do *not* define the class of competitor functions $\Theta$ to be the RKHS $\mathcal{H}$, but instead define $\Theta = \{u(\cdot) \,|\, u(x) = \sum_{t=1}^{T} \boldsymbol{\alpha}_t K(x, x_t),\, \boldsymbol{\alpha} \in \mathbb{R}^T\}$ — the set of kernel expansions over $\boldsymbol{x}_{\leq T}$. The model class is then $\mathcal{F} = \{P(\cdot|\boldsymbol{x}_{\leq T}, u(\cdot)) \,|\, u \in \Theta\}$. The representer theorem implies that competing against $\Theta$ is equivalent to competing against the RKHS.

It is easy to see that for this case, the NML density does not exist (recall Example 4.2) — the comparator class $\Theta$ contains very complex functions. However, the case is quite different for the penalized regret. Now let us consider using a GP prior. We choose $q$ to be the corresponding density over $\Theta$, which means that $q(u)$ is proportional to $\exp(-\|u\|_K^2/2)$, where $\|u\|_K^2 = \boldsymbol{\alpha}^T \boldsymbol{K} \boldsymbol{\alpha}$ with $\boldsymbol{K}_{i,j} = K(x_i, x_j)$ (recall Eq. 1). Now note that the penalty $-\log q(u)$ is just the RKHS norm $\|u\|_K^2/2$, up to an additive constant.

Using Theorem 4.3 and the equality in Theorem 3.2, we have the following corollary, which shows that the Bayesian strategy is precisely the NMAP distribution (for Gaussian regression).

**Corollary 4.4:** *For any $\boldsymbol{x}_{\leq T}$, in the Gaussian regression setting described above — where $\mathcal{F}$ and $\Theta$ are defined with respect to $\boldsymbol{x}_{\leq T}$ and where $q$ is the GP prior over $\Theta$ — we have that $P_{bayes}$ is a minimax strategy for the penalized regret, i.e. for all $\boldsymbol{y}_{\leq T}$, the regret $R_q(\boldsymbol{y}_{\leq T}, P_{bayes}, \Theta)$ does not exceed $R_q(\Theta)$. Furthermore, $P_{bayes}$ and $P_{map}$ are densities of the same distribution.*

Importantly, note that, while the competitor class $\mathcal{F}$ is constructed with full knowledge of $\boldsymbol{x}_{\leq T}$ in advance, the Bayesian strategy, $P_{\text{bayes}}$, can be implemented in an online manner in that it only needs to know $\boldsymbol{x}_{<t}$ for prediction at time $t$.

### Acknowledgments

We thank Manfred Opper and Manfred Warmuth for helpful discussions.

## Footnotes

[1]For these cases, the normalization constant of the NML distribution is not finite.

[2]To see this, simply observe that $P_{\mathrm{bayes}}(\boldsymbol{y}_{\leq T}) = \sum_\theta Q(\boldsymbol{y}_{\leq T}|\theta)q(\theta) \geq Q(\boldsymbol{y}_{\leq T}|\theta_{\mathrm{map}}(\boldsymbol{y}_{\leq T}))q(\theta_{\mathrm{map}}(\boldsymbol{y}_{\leq T}))$ and take the $-\log$ of both sides.

# References

K. S. Azoury and M. Warmuth. Relative loss bounds for on-line density estimation with the exponential family of distributions. *Machine Learning*, 43(3), 2001.

A. Barron, J. Rissanen, and B. Yu. The minimum description length principle in coding and modeling. *IEEE Trans. Information Theory*, 44, 1998.

Nicolo Cesa-Bianchi and Gabor Lugosi. Worst-case bounds for the logarithmic loss of predictors. *Machine Learning*, 43, 2001.

D. P. Foster. Prediction in the worst case. *Annals of Statistics*, 19, 1991.

D. P. Foster and R. A. Stine. The competitive complexity ratio. *Proceedings of 2001 Conf on Info Sci and Sys*, WP8, 2001.

P.D. Grunwald. A tutorial introduction to the minimum description length principle. *Advances in MDL: Theory and Applications*, 2005.

T. Hastie, R. Tibshirani, , and J. Friedman. *The Elements of Statistical Learning*. Springer, 2001.

S. M. Kakade and A. Y. Ng. Online bounds for bayesian algorithms. *Proceedings of Neural Information Processing Systems*, 2004.

M. Opper and D. Haussler. Worst case prediction over sequences under log loss. *The Mathematics of Information Coding, Extraction and Distribution*, 1998.

Y. Shtarkov. Universal sequential coding of single messages. *Problems of Information Transmission*, 23, 1987.

E. Takimoto and M. Warmuth. The minimax strategy for Gaussian density estimation. *Proc. 13th Annu. Conference on Comput. Learning Theory*, 2000.

Vladimir N. Vapnik. *Statistical Learning Theory*. Wiley, 1st edition, 1998.

V. Vovk. Competitive on-line statistics. *International Statistical Review*, 69, 2001.
